# Collective Inference on Markov Models for Modeling Bird Migration

**Daniel Sheldon**      **M. A. Saleh Elmohamed**      **Dexter Kozen**

Cornell University
Ithaca, NY 14853
{dsheldon,kozen}@cs.cornell.edu
saleh@cam.cornell.edu

## Abstract

We investigate a family of inference problems on Markov models, where many sample paths are drawn from a Markov chain and partial information is revealed to an observer who attempts to reconstruct the sample paths. We present algorithms and hardness results for several variants of this problem which arise by revealing different information to the observer and imposing different requirements for the reconstruction of sample paths. Our algorithms are analogous to the classical Viterbi algorithm for Hidden Markov Models, which finds the single most probable sample path given a sequence of observations. Our work is motivated by an important application in ecology: inferring bird migration paths from a large database of observations.

## 1 Introduction

Hidden Markov Models (HMMs) assume a generative model for sequential data whereby a sequence of states (or *sample path*) is drawn from a Markov chain in a hidden experiment. Each state generates an output symbol from alphabet $\Sigma$, and these output symbols constitute the data or *observations*. A classical problem, solved by the Viterbi algorithm, is to find the most probable sample path given certain observations for a given Markov model. We call this the *single path problem*; it is well suited to labeling or tagging a single sequence of data. For example, HMMs have been successfully applied in speech recognition [1], natural language processing [2], and biological sequencing [3].

We introduce two generalizations of the single path problem for performing *collective inference* on Markov models, motivated by an effort to model bird migration patterns using a large database of static observations. The eBird database hosted by the Cornell Lab of Ornithology contains millions of bird observations from throughout North America, reported by the general public using the eBird web application.[1] Observations report location, date, species and number of birds observed. The eBird data set is very rich; the human eye can easily discern migration patterns from animations showing the observations as they unfold over time on a map of North America.[2] However, the eBird data are *static*, and they do not explicitly record movement, only the distributions at different points in time. Conclusions about migration patterns are made by the human observer. Our goal is to build a mathematical framework to infer dynamic migration models from the static eBird data. Quantitative migration models are of great scientific and practical import: for example, this problem arose out of an interdisciplinary project at Cornell University to model the possible spread of avian influenza in North America through wild bird migration.

The migratory behavior for a species of birds can be modeled using a single generative process that independently governs how individual birds fly between locations, giving rise to the following

inference problem: a hidden experiment simultaneously draws many independent sample paths from a Markov chain, and the observations reveal aggregate information about the collection of sample paths at each time step, from which the observer attempts to reconstruct the paths. For example, the eBird data estimate the geographical distribution of a species on successive days, but do not track individual birds.

We discuss two problems within this framework. In the *multiple path problem*, we assume that exactly $M$ independent sample paths are drawn from the Markov model, and the observations reveal the number of paths that output symbol $\alpha$ at time $t$, for each $\alpha$ and $t$. The observer seeks the most likely collection of paths given the observations. The *fractional path problem* is a further generalization in which paths are divisible entities. The observations reveal the fraction of paths that output symbol $\alpha$ at time $t$, and the observer's job is to find the most likely (in a sense to be defined later) weighted collection of paths given the observations. Conceptually, the fractional path problem can be derived from the multiple path problem by letting $M$ go to infinity; or it has a probabilistic interpretation in terms of distributions over paths.

After discussing some preliminaries in section 2, sections 3 and 4 present algorithms for the multiple and fractional path problems, respectively, using network flow techniques on the *trellis graph* of the Markov model. The multiple path problem in its most general form is NP-hard, but can be solved as an integer program. The special case when output symbols uniquely identify their associated states can be solved efficiently as a flow problem; although the single path problem is trivial in this case, the multiple and fractional path problems remain interesting. The fractional path problem can be solved by linear programming. We also introduce a practical extension to the fractional path problem, including slack variables allowing the solution to deviate slightly from potentially noisy observations. In section 5, we demonstrate our techniques with visualizations for the migration of *Archilochus colubris*, the Ruby-throated Hummingbird, devoting some attention to a challenging problem we have neglected so far: estimating species distributions from eBird observations.

We briefly mention some related work. Caruana et al. [4] and Phillips et al. [5] used machine learning techniques to model bird distributions from observations and environmental features. For problems on sequential data, many variants of HMMs have been proposed [3], and recently, conditional random fields (CRFs) have become a popular alternative [6]. Roth and Yih [7] present an integer programming inference framework for CRFs that is similar to our problem formulations.

## 2 Preliminaries

### 2.1 Data Model and Notation

A Markov model $(V, p, \Sigma, \sigma)$ is a Markov chain with state set $V$ and transition probabilities $p(u, v)$ for all $u, v \in V$. Each state generates a unique output symbol from alphabet $\Sigma$, given by the mapping $\sigma : V \to \Sigma$. Although some presentations allow each state to output multiple symbols with different emission probabilities, we lose no generality assuming that each state emits a unique symbol — to encode a model where state $v$ output multiple symbols, we simply duplicate $v$ for each symbol and encode the emission probabilities into the transitions. Of course, $\sigma$ need not be one-to-one. It is useful to think of $\sigma$ as a partition of the states, letting $V_\alpha = \sigma^{-1}(\alpha)$ be the set of all states that output $\alpha$. We assume each model has a distinguished start state $s$ and output symbol `start`.

Let $\mathcal{Y} = V^T$ be the set of all possible sample paths of length $T$. We represent a path $\mathbf{y} \in \mathcal{Y}$ as a row vector $\mathbf{y} = (y_1, \ldots, y_T)$, and a collection of $M$ paths as the $M \times T$ matrix $\mathbf{Y} = (y_{it})$, with each row $\mathbf{y}_{i\cdot}$ representing an independent sample path. The transition probabilities induce a distribution $\lambda$ on $\mathcal{Y}$, where $\lambda(\mathbf{y}) = \prod_{t=1}^{T-1} p(y_t, y_{t+1})$. We will also consider arbitrary distributions $\pi$ over $\mathcal{Y}$, letting $Y = (Y_1, \ldots, Y_T)$ denote a random path from $\pi$. Then, for example, we write $\Pr_\pi [Y_t = u]$ to be the probability under $\pi$ that the $t$th state is $u$, and $E_\pi [f(Y)]$ to be the expected value of $f(Y)$ for any function $f$ of a random path $Y$ drawn from $\pi$. Note that $\mathbf{Y}$ (boldface) denotes a matrix of $M$ paths, while $Y$ denotes a random path.

### 2.2 The Trellis Graph and Viterbi as Shortest Path

To develop our flow-based algorithms, it is instructive to build upon a shortest-path interpretation of the Viterbi algorithm [7]. In an instance of the single path problem we are given a model $(V, p, \Sigma, \sigma)$

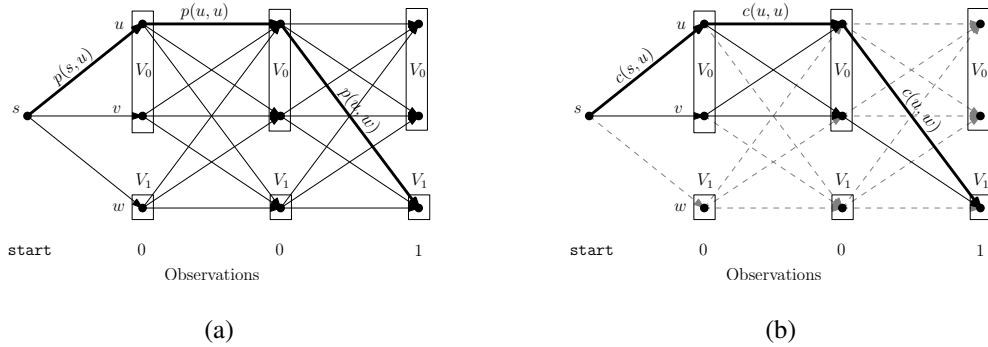

Figure 1: Trellis graph for Markov model with states $\{s, u, v, w\}$ and alphabet $\{\texttt{start}, 0, 1\}$. States $u$ and $v$ output the symbol 0, and state $w$ outputs the symbol 1. (a) The bold path is feasible for the specified observations, with probability $p(s, u)p(u, u)p(u, w)$. (b) Infeasible edges have been removed (indicated by light dashed lines), and probabilities changed to costs. The bold path has cost $c(s, u) + c(u, u) + c(u, w)$.

and observations $\alpha_1, \ldots, \alpha_T$, and we seek the most probable path $\mathbf{y}$ given these observations. We call path $\mathbf{y}$ *feasible* if $\sigma(y_t) = \alpha_t$ for all $t$; then we wish to maximize $\lambda(\mathbf{y})$ over feasible $\mathbf{y}$. The problem is conveniently illustrated using the *trellis graph* of the Markov model (Figure 1). Here, the states are replicated for each time step, and edges connect a state at time $t$ to its possible successors at time $t + 1$, labeled with the transition probability. A feasible path must pass through partition $V_{\alpha_t}$ at step $t$, so we can prune all edges incident on other partitions, leaving only feasible paths. By defining the cost of an edge as $c(u, v) = -\log p(u, v)$, and letting the path cost $c(\mathbf{y})$ be the sum of its edge costs, straightforward algebra shows that $\arg\max_{\mathbf{y}} \lambda(\mathbf{y}) = \arg\min_{\mathbf{y}} c(\mathbf{y})$, i.e., the path of maximum probability becomes the path of minimum cost under this transformation. Thus the Viterbi algorithm finds the shortest feasible path in the trellis using edge lengths $c(u, v)$.

## 3 Multiple Path Problem

In the multiple path problem, $M$ sample paths are drawn from the model and the observations reveal the number of paths $N_t(\alpha)$ that output $\alpha$ at time $t$, for all $\alpha$ and $t$; or, equivalently, the multiset $A_t$ of output symbols at time $t$. The objective is to find the most probable collection $\mathbf{Y}$ that is feasible, meaning it produces multisets $A_1, \ldots, A_T$. The probability $\lambda(\mathbf{Y})$ is just the product of the path-wise probabilities:

$$\lambda(\mathbf{Y}) = \prod_{i=1}^{M} \lambda(\mathbf{y}_i) = \prod_{i=1}^{M} \prod_{t=1}^{T-1} p(y_{i,t}, y_{i,t+1}). \tag{1}$$

Then the formal specification of this problem is

$$\max_{\mathbf{Y}} \lambda(\mathbf{Y}) \text{ subject to } |\{i : y_{i,t} \in V_\alpha\}| = N_t(\alpha) \text{ for all } \alpha, t. \tag{2}$$

### 3.1 Reduction to the Single Path Problem

A naive approach to the multiple path problem reduces it to the single path problem by creating a new Markov model on state set $V^M$ where state $\langle v_1, \ldots, v_M \rangle$ encodes an entire tuple of original states, and the transition probabilities are given by the product of the element-wise transition probabilities:

$$p(\langle u_1, \ldots, u_M \rangle, \langle v_1, \ldots, v_M \rangle) = \prod_{i=1}^{M} p(u_i, v_i).$$

A state from the product space $V^M$ corresponds to an entire column of the matrix $\mathbf{Y}$, and by changing the order of multiplication in (1), we see that the probability of a path in the new model is equal to the probability of the entire collection of paths in the old model. To complete the reduction, we form a new alphabet $\hat{\Sigma}$ whose symbols represent multisets of size $M$ on $\Sigma$. Then the solution to (2) can be found by running the Viterbi algorithm to find the most likely sequence of states from $V^M$ that produce output symbols (multisets) $A_1, \ldots, A_T$. The running time is polynomial in $|V^M|$ and $|\hat{\Sigma}|$, but exponential in $M$.

## 3.2 Graph Flow Formulation

Can we do better than the naive approach? Viewing the cost of a path as the cost of routing one unit of flow along that path in the trellis, a minimum cost collection of $M$ paths is equivalent to a minimum cost flow of $M$ units through the trellis — given $M$ paths, we can route one unit along each to get a flow, and we can decompose any flow of $M$ units into paths each carrying a single unit of flow. Thus we can write the optimization problem in (2) as the following flow integer program, with additional constraints that the flow paths generate the correct observations. The decision variable $x_{uv}^t$ indicates the flow traveling from $u$ to $v$ at time $t$; or, the number of sample paths that transition from $u$ to $v$ at time $t$.

$$\min \sum_{u,v,t} c(u,v) x_{uv}^t$$

(IP)
$$\text{s.t.} \quad \sum_u x_{uv}^t = \sum_w x_{vw}^{t+1} \qquad \text{for all } v, t, \qquad (3)$$

$$\sum_{u \in V_\alpha, v \in V} x_{uv}^t = N_t(\alpha) \qquad \text{for all } \alpha, t, \qquad (4)$$

$$x_{uv}^t \in \mathbb{N} \qquad \text{for all } u, v, t.$$

The flow conservation constraints (3) are standard: the flow into $v$ at time $t$ is equal to the flow leaving $v$ at time $t+1$. The observation constraints (4) specify that $N_t(\alpha)$ units of flow leave partition $V_\alpha$ at time $t$. These also imply that exactly $M$ units of flow pass through each level of the trellis, by summing over all $\alpha$,

$$\sum_{u,v} x_{uv}^t = \sum_\alpha \sum_{u \in V_\alpha, v \in V} x_{uv}^t = \sum_\alpha N_t(\alpha) = M.$$

Without the observation constraints, IP would be an instance of the minimum-cost flow problem [8], which is solvable in polynomial time by a variety of algorithms [9]. However, we cannot hope to encode the observation constraints into the flow framework, due to the following result.

**Lemma 1.** *The multiple path problem is NP-hard.*

The proof of Lemma 1 is by reduction from SET COVER, and is omitted. One may use a general purpose integer program solver to solve IP directly; this may be efficient in some cases despite the lack of polynomial time performance guarantees. In the following sections we discuss alternatives that are efficiently solvable.

## 3.3 An Efficient Special Case

In the special case when $\sigma$ is one-to-one, the output symbols uniquely identify their generating states, so we may assume that $\Sigma = V$, and the output symbol is always the name of the current state. To see how the problem IP simplifies, we now have $V_u = \{u\}$ for all $u$, so each partition consists of a single state, and the observations completely specify the flow through each node in the trellis:

$$\sum_v x_{uv}^t = N_t(u) \quad \text{for all } u, t. \qquad (4')$$

Substituting the new observation constraints (4′) for time $t+1$ into the RHS of the flow conservation constraints (3) for time $t$ yield the following replacements:

$$\sum_u x_{uv}^t = N_{t+1}(v) \quad \text{for all } v, t. \qquad (3')$$

This gives an equivalent set of constraints, each of which refers only to variables $x_{uv}^t$ for a single $t$. Hence the problem can be decomposed into $T-1$ disjoint subproblems for $t = 1, \ldots, T-1$. The $t$th subproblem $\text{IP}_t$ is given in Figure 2(a), and illustrated on the trellis in Figure 2(b). State $u$ at time $t$ has a supply of $N_t(u)$ units of flow coming from the previous step, and we must route $N_{t+1}(v)$ units of flow to state $v$ at time $t+1$, so we place a demand of $N_{t+1}(v)$ at the corresponding node. Then the problem reduces to finding a minimum cost routing of the supply from time $t$ to meet the demand at time $t+1$, solved separately for all $t = 1, \ldots, T-1$. The problem $\text{IP}_t$ an instance of the transportation problem [10], a special case of the minimum-cost flow problem. There are a variety of efficient algorithms to solve both problems [8,9], or one may use a general purpose linear program (LP) solver; any basic solution to the LP relaxation of $\text{IP}_t$ is guaranteed to be integral [8].

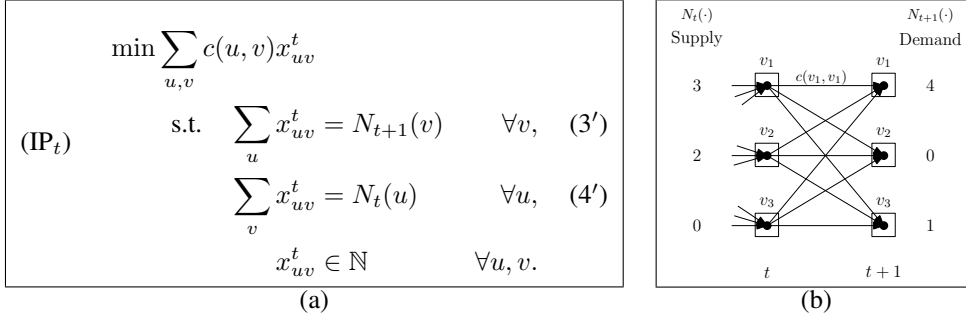

$$\text{(IP}_t) \quad \min \sum_{u,v} c(u,v) x_{uv}^t$$

$$\text{s.t.} \quad \sum_u x_{uv}^t = N_{t+1}(v) \qquad \forall v, \quad (3')$$

$$\sum_v x_{uv}^t = N_t(u) \qquad \forall u, \quad (4')$$

$$x_{uv}^t \in \mathbb{N} \qquad \forall u, v.$$

(a)

(b)

Figure 2: (a) The definition of subproblem $\text{IP}_t$. (b) Illustration on the trellis.

## 4   Fractional Path Problem

In the fractional path problem, a path is a divisible entity. The observations specify $q_t(\alpha)$, the fraction of paths that output $\alpha$ at time $t$, and the observer chooses $\pi(\mathbf{y})$ fractional units of each path $\mathbf{y}$, totaling one unit, such that $q_t(\alpha)$ units output $\alpha$ at time $t$. The objective is to maximize $\prod_{\mathbf{y}\in\mathcal{Y}} \lambda(\mathbf{y})^{\pi(\mathbf{y})}$. Put another way, $\pi$ is a distribution over paths such that $\Pr_\pi[Y_t \in V_\alpha] = q_t(\alpha)$, i.e., $q_t$ specifies the marginal distribution over symbols at time $t$. By taking the logarithm, an equivalent objective is to maximize $E_\pi[\log \lambda(Y)]$, so we seek the distribution $\pi$ that maximizes the expected log-probability of a path $Y$ drawn from $\pi$. Conceptually, the fractional path problem arises by letting $M \to \infty$ in the multiple path problem and normalizing to let $q_t(\alpha) = N_t(\alpha)/M$ specify the fraction of paths that output $\alpha$ at time $t$. Operationally, the fractional path problem is modeled by the LP relaxation of IP, which routes one splittable unit of flow through the trellis.

$$\min \sum_{u,v,t} c(u,v) x_{uv}^t$$

$$\text{(RELAX)} \quad \text{s.t.} \quad \sum_u x_{uv}^t = \sum_w x_{vw}^{t+1} \qquad \text{for all } v, t,$$

$$\sum_{u\in V_\alpha} \sum_{v\in V} x_{uv}^t = q_t(\alpha) \qquad \text{for all } \alpha, t, \qquad (5)$$

$$x_{uv}^t \geq 0 \qquad \text{for all } u, v, t.$$

It is easy to see that a unit flow $x$ corresponds to a probability distribution $\pi$. Given any distribution $\pi$, let $x_{uv}^t = \Pr_\pi[Y_t = u, Y_{t+1} = v]$; then $x$ is a flow because the probability a path enters $v$ at time $t$ is equal to the probability it leaves $v$ at time $t+1$. Conversely, given a unit flow $x$, any path decomposition assigning flow $\pi(\mathbf{y})$ to each $\mathbf{y} \in \mathcal{Y}$ is a probability distribution because the total flow is one. In general, the decomposition is not unique, but any choice yields a distribution $\pi$ with the same objective value. Furthermore, under this correspondence, $x$ satisfies the marginal constraints (5) if and only if $\pi$ has the correct marginals:

$$\sum_{u\in V_\alpha} \sum_{v\in V} x_{uv}^t = \sum_{u\in V_\alpha} \sum_{v\in V} \Pr[Y_t = u, Y_{t+1} = v] = \sum_{u\in V_\alpha} \Pr[Y_t = u] = \Pr[Y_t \in V_\alpha].$$

Finally, we can rewrite the objective function in terms of paths:

$$\sum_{u,v,t} c(u,v) x_{uv}^t = \sum_{\mathbf{y}\in\mathcal{Y}} \pi(\mathbf{y}) c(\mathbf{y}) = E_\pi[c(Y)] = E_\pi[-\log \lambda(Y)].$$

By switching signs and changing from minimization to maximization, we see that RELAX solves the fractional path problem. This problem is very similar to maximum entropy or minimum cross entropy modeling, but the details are slightly different: such a model would typically find the distribution $\pi$ with the correct marginals that minimizes the cross entropy or Kullback-Leibler divergence [11] between $\lambda$ and $\pi$, which, after removing a constant term, reduces to minimizing $E_\lambda[-\log \pi(Y)]$. Like IP, the RELAX problem also decomposes into subproblems in the case when $\sigma$ is one-to-one, but this simplification is incompatible with the slack variables introduced in the following section.

## 4.1 Incorporating Slack

In our application, the marginal distributions $q_t(\cdot)$ are themselves estimates, and it is useful to allow the LP to deviate slightly from these marginals to find a better overall solution. To accomplish this, we add slack variables $\delta_u^t$ into the marginal constraints (5), and charge for the slack in the objective function. The new marginal constraints are

$$\sum_{u \in V_\alpha} \sum_{v \in V} x_{uv}^t = q_t(\alpha) + \delta_\alpha^t \qquad \text{for all } \alpha, t, \tag{5'}$$

and we add the term $\sum_{\alpha,t} \gamma_\alpha^t |\delta_\alpha^t|$ into the objective function to charge for the slack, using a standard LP trick [8] to model the absolute value term. The slack costs $\gamma_\alpha^t$ can be tailored to individual input values; for example, one may want to charge more to deviate from a confident estimate. This will depend on the specific application. We also add the necessary constraints to ensure that the new marginals $q_t'(\alpha) = q_t(\alpha) + \delta_\alpha^t$ form a valid probability distribution for all $t$.

## 5  Demonstration

In this section, we demonstrate our techniques by using the fractional path problem to create visualizations showing likely migration routes of *Archilochus colubris*, the Ruby-throated Hummingbird, a common bird whose range is relatively well covered by eBird observations. We work in discretized space and time, dividing the map into grid cells and the year into weeks. We must specify the Markov model governing transitions between locations (grid cells) in successive weeks; also, we require estimates $q_t(\cdot)$ for the weekly distributions of hummingbirds across locations. Since the actual eBird observations are highly non-uniform in space and time, estimating weekly distributions requires significant inference for locations with few or no observations. In the appendix, we outline one approach based on *harmonic energy minimization* [12], but we may use any technique that produces weekly distributions $q_t(u)$ and slack costs $\gamma_u^t$. Improving these estimates, say, by incorporating important side information such as climate and habitat features, could significantly improve the overall model. Finally, although our final observations $q_t(\cdot)$ are distributions over states (locations) and not output symbols — i.e., $\sigma$ is one-to-one — we cannot use the simplification from section 3.3 because we incorporate slack into the model.

### 5.1  eBird Data

Launched in 2002, eBird is a *citizen science* project run by the Cornell Lab of Ornithology, leveraging the data gathering power of the public. On the eBird website, birdwatchers submit checklists of birds they observe, indicating a count for each species, along with the location, date, time and additional information. Our data set consists of the $428{,}648$ *complete* checklists from 1995[3] through 2007, meaning the reporter listed all species observed. This means we can infer a count of zero, or a *negative observation*, for any species not listed. Using a land cover map from the United States Geological Survey (USGS), we divide North America into grid cells that are roughly 225 km on a side. All years of data are aggregated into one, and the year is divided into weeks so $t = 1, \ldots, 52$ represents the week of the year.

### 5.2  Migration Inference

Given weekly distributions $q_t(u)$ and slack costs $\gamma_u^t$ (see the appendix), it remains to specify the Markov model. We use a simple Gaussian model favoring short flights, letting $p(u, v) \propto \exp(-d(u, v)^2/\sigma^2)$, where $d(u, v)$ measures the distance between grid cell centers. This corresponds to a squared distance cost function. To reduce problem size, we omitted variables $x_{uv}^t$ from the LP when $d(u, v) > 1350$ km, effectively setting $p(u, v) = 0$. We also found it useful to impose upper bounds $\delta_u^t \leq q_t(u)$ on the slack variables so no single value could increase by more than a factor of two. Our final LP, which was solved using the MOSEK optimization toolbox, had 78,521 constraints and 3,031,116 variables.

Figure 3 displays the migration paths our model inferred for the four weeks starting on the dates indicated. The top row shows the distribution and paths inferred by the model; grid cells colored

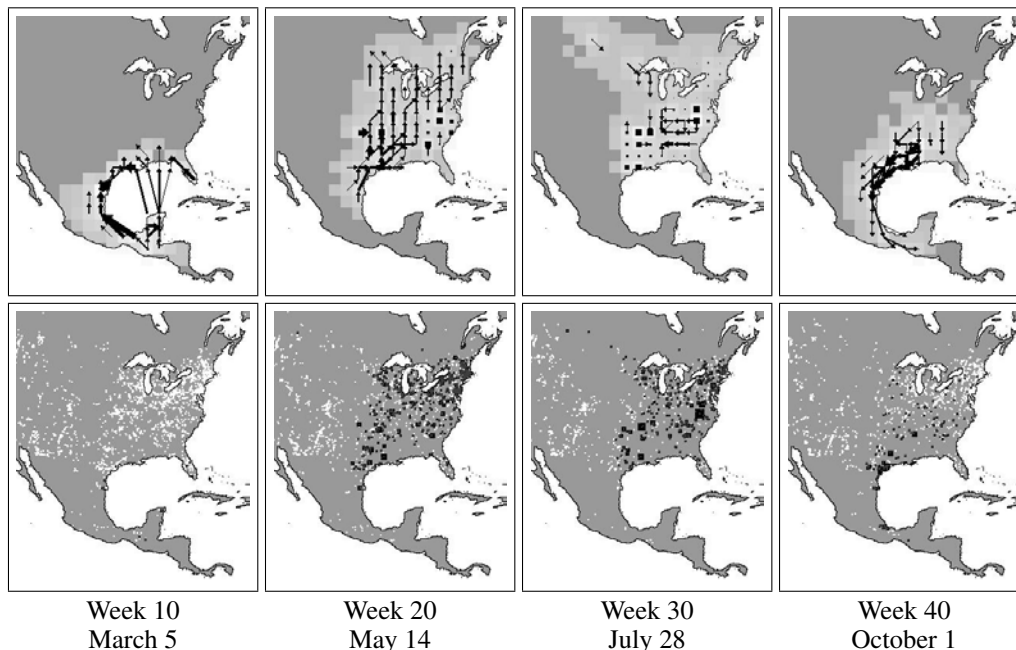

| Week 10 | Week 20 | Week 30 | Week 40 |
| March 5 | May 14 | July 28 | October 1 |

Figure 3: Ruby-throated Hummingbird migration. See text for description.

in lighter shades have more birds (higher values for $q'_t(u)$). Arrows indicate flight paths $(x^t_{uv})$ between the week shown and the following week, with line width proportional to flow $x^t_{uv}$. In the bottom row, the raw data is given for comparison. White dots indicate negative observations; black squares indicate positive observations, with size proportional to count. Locations with both positive and negative observations appear a charcoal color. The inferred distributions and paths are consistent with both seasonal ranges and written accounts of migration routes. For example, in the summary paragraph on migration from the *Archilochus colubris* species account in Birds of North America [13], Robinson et al. write "Many fly across Gulf of Mexico, but many also follow coastal route. Routes may differ for north- and southbound birds."

## Acknowledgments

We are grateful to Daniel Fink, Wesley Hochachka and Steve Kelling from the Cornell Lab of Ornithology for useful discussions. This work was supported in part by ONR Grant N00014-01-1-0968 and by NSF grant CCF-0635028. The views and conclusions herein are those of the authors and do not necessarily represent the official policies or endorsements of these organizations or the US Government.

## Footnotes

[1] http://ebird.org

[2] http://www.avianknowledge.net/visualization

[3]Users may enter historical observations.

## References

[1] L. R. Rabiner. A tutorial on hidden Markov models and selected applications in speech recognition. *Proceedings of the IEEE*, 77(2):257–286, 1989.

[2] E. Charniak. Statistical techniques for natural language parsing. *AI Magazine*, 18(4):33–44, 1997.

[3] R. Durbin, S. Eddy, A. Krogh, and G. Mitchison. *Biological sequence analysis: Probabilistic models of proteins and nucleic acids*. Cambridge University Press, 1998.

[4] R. Caruana, M. Elhaway, A. Munson, M. Riedewald, D. Sorokina, D. Fink, W. M. Hochachka, and S. Kelling. Mining citizen science data to predict prevalence of wild bird species. In *SIGKDD*, 2006.

[5] S. J. Phillips, M. Dudík, and R. E. Schapire. A maximum entropy approach to species distribution modeling. In *ICML*, 2004.

[6] J. Lafferty, A. McCallum, and F. Pereira. Conditional random fields: Probabilistic models for segmenting and labeling sequence data. *ICML*, 2001.

[7] D. Roth and W. Yih. Integer linear programming inference for conditional random fields. *ICML*, 2005.

[8] V. Chvátal. *Linear Programming*. W.H. Freeman, New York, NY, 1983.

[9] A. V. Goldberg, S. A. Plotkin, and E. Tardos. Combinatorial algorithms for the generalized circulation problem. *Math. Oper. Res.*, 16(2):351–381, 1991.

[10] G. B. Dantzig. Application of the simplex method to a transportation problem. In T. C. Koopmans, editor, *Activity Analysis of Production and Allocation*, volume 13 of *Cowles Commission for Research in Economics*, pages 359–373. Wiley, 1951.

[11] J. Shore and R. Johnson. Properties of cross-entropy minimization. *IEEE Trans. on Information Theory*, 27:472–482, 1981.

[12] X. Zhu, Z. Ghahramani, and J. Lafferty. Semi-supervised learning using Gaussian fields and harmonic functions. In *ICML*, 2003.

[13] T. R. Robinson, R. R. Sargent, and M. B. Sargent. Ruby-throated Hummingbird (*Archilochus colubris*). In A. Poole and F. Gill, editors, *The Birds of North America*, number 204. The Academy of Natural Sciences, Philadelphia, and The American Ornithologists' Union, Washington, D.C., 1996.

[14] D. Aldous and J. Fill. *Reversible Markov Chains and Random Walks on Graphs*. Monograph in Preparation, http://www.stat.berkeley.edu/users/aldous/RWG/book.html.

# A    Estimating Weekly Distributions from eBird

Our goal is to estimate $q_t(u)$, the fraction of birds in grid cell $u$ during week $t$. Given enough observations, we can estimate $q_t(u)$ using the average number of birds counted per checklist, a quantity we call the *rate* $r_t(u)$. However, even for a bird with good eBird coverage, there are cells with few or no observations during some weeks. To fill these gaps, we use the *harmonic energy minimization* technique [12] to determine values for empty cells based on neighbors in space and time. This technique uses a graph-based similarity structure, in our case the 3-dimensional lattice built on points $u_t$, where $u_t$ represents cell $u$ during week $t$. Edges are weighted, with weights representing similarity between points. Point $u_t$ is connected to its four grid neighbors in time slice $t$ by edges of unit weight, excluding edges between cells separated by water (specifically, when the line connecting the centers is more than half water). Point $u_t$ is also connected to points $u_{t-1}$ and $u_{t+1}$ with weight $1/4$, to achieve some temporal smoothing.

Harmonic energy minimization learns a function $f$ on the graph; the idea is to match $r_t(u)$ on points with sufficient data and find values for other points according to the similarity structure. To this end, we designate some boundary points for which the value of $f$ is fixed by the data, while other points are interior points. The value of $f$ at interior point $u_t$ is determined by the expected value of the following random experiment: perform a random walk starting from $u_t$, following outgoing edges with probability proportional to their weight. When the walk first hits a boundary point $v_{t'}$, terminate and accept the boundary value $f(v_{t'})$. In this way, the values at interior points are a weighted average of nearby boundary values, where "nearness" is interpreted as the absorption probability in an absorbing random walk. We derive a measure of confidence in the value $f(u_t)$ from the same experiment: let $h(u_t)$ be the expected number of steps for the random walk from $u_t$ to hit the boundary (the *hitting time* of the boundary set [14]). When $h(u_t)$ is small, $u_t$ is close to the boundary and we are more confident in $f(u_t)$.

Rather than choosing a threshold on the number of observations required to be a boundary point, we create a soft boundary by designating all points $u_t$ as interior points, and adding one boundary node to the graph structure for each observation, connected by an edge of unit weight to the cell in which it occurred, with value equal to the number of birds observed. As point $u_t$ gains more observations, its behavior approaches that of a hard boundary: with probability approaching one, the walk from $u_t$ will reach an observation in the first step, so $f(u_t)$ will approach $r_t(u)$, the average of the observations. As a conservative measure, each node is also connected to a sink with boundary value 0, to prevent values from propagating over very long distances.

We compute $h$ and $f$ iteratively using standard techniques. Since $f(u_t)$ approximates the rate $r_t(u)$, we multiply by the land mass of cell $u$ to get an estimate $\hat{q}_t(u)$ for the (relative) number of birds in cell $u$ at time $t$. Finally, we normalize $\hat{q}$ for each time slice $t$, taking $q_t(u) = \hat{q}_t(u)/\sum_u \hat{q}_t(u)$. For slack costs, we set $\gamma_u^t = \gamma_0/h(u_t)$ to be inversely proportional to boundary hitting time, with $\gamma_0 \approx 261$ chosen in conjunction with the transition costs in section 5.2 so the average cost for a unit of slack is the same as moving 600 km.

